# Load and Attentional Bayes

**Peter Dayan**
Gatsby Computational Neuroscience Unit, UCL
London, England, WC1N 3AR
dayan@gatsby.ucl.ac.uk

## Abstract

Selective attention is a most intensively studied psychological phenomenon, rife with theoretical suggestions and schisms. A critical idea is that of limited capacity, the allocation of which has produced continual conflict about such phenomena as *early* and *late* selection. An influential resolution of this debate is based on the notion of perceptual load (Lavie, 2005), which suggests that low-load, easy tasks, because they underuse the total capacity of attention, mandatorily lead to the processing of stimuli that are irrelevant to the current attentional set; whereas high-load, difficult tasks grab all resources for themselves, leaving distractors high and dry. We argue that this theory presents a challenge to Bayesian theories of attention, and suggest an alternative, statistical, account of key supporting data.

## 1 Introduction

It was some fifty years after James (1950)'s famously poetic description of our capacities for attention that more analytically-directed experiments began, based originally on dichotic listening Cherry (1953). There are three obvious dichotic tasks: (i) being able to interpret fully two *separate* streams of information coming into the two ears; (ii) the less ambitious version of this of being able to interpret fully *one* of the streams, specified top-down, without interference from the other one; and (iii) being able to *combine* information from the two ears appropriately, perhaps into a single percept. Various forms, interpretations and conflicts about these three tasks have permeated the field of attention ever since (Driver, 2001; Paschler, 1998), driven by different notions of the computational tasks and constraints at hand.

The experiments in dichotic listening coincided with the quickly burgeoning realization that mathematical concepts from Shannonian information theory would be very helpful for understanding biological information processing. One central concept in information theory is that of a limited capacity channel, and Broadbent (1958) adopted this as a formal basis for understanding the necessity for, and hence the nature of, selection. Broadbent (1958)'s theory critically involves *early* selection, in that following a first, automatic, parallel stage of low-level perceptual processing (itself the subject of important studies of bottom-up influences on selection, Zhaoping, 2006), a relevant stream should be selected for subsequent higher-level, semantic, processing, leaving any irrelevant streams in the cold. However, evidence that information in unattended streams is actually processed semantically (*eg* being able to bias the perception of ambiguous words in the attended stream; Mackay, 1973), led to alternative theories, either *late* selection (influentially, Deutsch and Deutsch, 1963; Duncan, 1980), in which both streams are fully processed, but with the irrelevant stream being prevented by a selective process at the last step from entering memory or awareness, or weaker forms of this, such as the notion that elements from the irrelevant stream might be *attenuated*, only sometimes progressing through to higher levels of processing (Treisman, 1960, 1969). Many hypotheses in the field depend on this collection of metaphors, nicely exemplified by the zoom-lens theory of Eriksen and St. James (1986) (based on influential experiments on distractor processing such as Eriksen and Eriksen, 1974), which suggests that the smaller the attentional focus, the more intense it can somehow be, given that the limited capacity is 'spread' over a smaller area.

However, of course, late selection makes little sense from a limited capacity viewpoint; and short of a theory of what controls the degree of attenuation of irrelevant stimuli, Treisman (1960)'s idea is hard to falsify. Here, we consider the seminal sharp operationalization of Lavie and Tsal (1994); Lavie (2005), who suggested that attenuation is a function of *load*, such that in easy tasks, irrelevant data is *always* processed, even at the cost of worse performance on the relevant information, whereas in difficult tasks, no capacity remains, and so distractors are more effectively removed. To reiterate, the attentional load hypothesis, although an attractive formalization of attenuation, suggests that the brain is unable on *easy* tasks to exclude information that is known to be irrelevant. It therefore involves an arguably infelicitous combination of sophisticated attentional shaping (as to what can be attended in high-load situations) with inept control.

Although the Bayesian revolution in cognitive science has had a huge impact over modern views of sensory processing (see, for instance, Rao et al., 2002, and references therein), having the ability to resolve many issues in the field as a whole, there are few recent attempts to build probabilistic models for selective attention (see Shaw, 1982; Palmer, 1994; Dayan and Zemel, 1999; Navalpakkam and Itti, 2006; Mozer and Baldwin, 2008; Yu and Dayan, 2005; Yu et al., 2008). This is despite the many other computational models of attention (see Itti and Koch, 2001; Zhaoping, 2006). Indeed, Whiteley and Sahani (2008) have suggested that this lacuna arises from a focus on optimal Bayesian inference in the face of small numbers of objects in the focus of attention, rather than the necessity of using approximate methods in the light of realistic, cluttered, complex scenes.

Some of the existing probabilistic models are aimed at variants of search (Navalpakkam and Itti, 2006; Mozer and Baldwin, 2008); however others, including Palmer (1994); Dayan and Zemel (1999), and one of the two models in Yu et al. (2008), are more similar to the account here. They acknowledge that there is a critical limited resource coming from the existence of neurons with large receptive fields into which experimenters slot multiple sensory objects, some relevant, some irrelevant. Probabilistically-correct inference should then implement selection, when data that is *known* to be irrelevant is excluded to the advantage of the relevant information (*eg* Dayan and Zemel, 1999; Palmer, 1994). However, in other circumstances, it will be appropriate to take advantage of the information about the target that is available in the neurons with large fields, even if this means allowing some influence on the final decisions from distractors.

Here, we build a Bayesian-inspired account of key data used to argue for the attentional load hypothesis (based on an extension of Yu et al. (2008)'s model of Eriksen and Eriksen (1974)). Section 2 describes the key data; section 3 the model and results; and section 4 discusses the implications.

## 2   Attentional Load

Figure 1 shows the central experiment and results from Lavie and de Fockert (2003) that we set out to capture. Subjects had to report the identity of a target letter that was either an 'X' or an 'N' (here, the former) presented in one of eight locations arranged in a circle around the fixation point. The reaction times and accuracies of their selections were measured. There was also a distractor letter in the further periphery (the larger 'N') which was either *compatible* (*ie* the same as the target), *incompatible* (as here, the opposite of the target), or, in so-called *neutral* trials, a different letter altogether.

Figure 1A-C show the three key conditions. Figure 1A is a high-load condition, in that there are irrelevant non-targets in the remaining 7 positions around the circle. Figure 1B is a low-load condition, since there is no non-target. Figure 1C is a critical control, called the degraded low-load condition, and was actually the main topic of Lavie and de Fockert (2003). In this, the *difficulty* of the sensory processing was increased (by making the target smaller and dimmer) without changing the attentional (*ie* selectional) load.

Figure 1D shows the mean reaction times (RTs) for these conditions for the three sorts of distractor (RTs suffice here, since there was no speed accuracy tradeoff at work in the different conditions; data not shown). There are three key results:

1. The central finding about attentional load is that the distractor exerted a significant effect over target processing *only* in the low load case – that is, an incompatible distractor slowed down the RTs compared with a neutral distractor for the low load case but not the high load case.

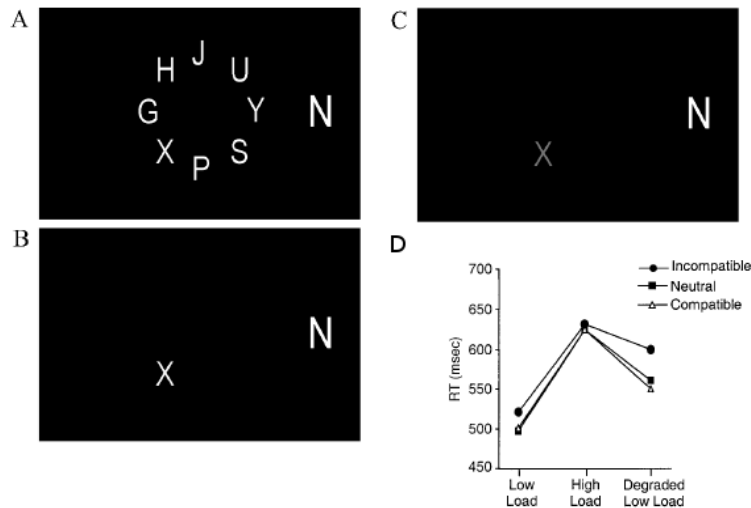

Figure 1: The attentional load task, from Lavie and de Fockert (2003). Subjects had to judge whether a target letter in the central circle around fixation was 'N' or 'X' in the face of a compatible, incompatible (shown) or neutral distractor. A) high-load condition with non-target letters occupying the other positions in the circle. B) low-load condition with no non-target letters. C) degraded low-load condition with no non-targets but a smaller (not shown) and darker target. D) reaction times (RTs) for the conditions, averaging only over correct choices.

2. Since, in the degraded low-load case the RTs were slower but the influence of the distractor was if anything *greater*, this could not just be a function of the processing time or difficulty. Indeed, Lavie and de Fockert (2003) noted the distinction made by Norman and Bobrow (1975) between data- and resource-limited processing, with excess resources (putatively ample, given the low load) unable to make up for the poor quality sensory data, and so predicted this greater distractor impact.

3. It is apparent that compatible distractors were of almost no help in any case, whereas incompatible distractors were harmful.

## 3   The Bayesian model

The data in figure 1 pose the question for normative modeling as to why the distractor would corrupt processing of the target in the easy, low-load, case, but not the difficult, high-load case. No normative account could simply assume that extra data 'leak' through in the low-load condition (which is the attentional load hypothesis) if the subjects have the ability to fashion attention far more finely in other cases, such as that of high load.

We argue that these results stem from the simple observation that the visual system has available receptive fields with a range of sizes, including smaller, spatially precise ones, which can be nicely confined to the target; and larger, spatially extended ones, which may include both target and distractor. In this case, normative processing will combine information from all the receptive fields, with Bayesian inference and marginalization exactly eliminating any substantial impact from those that are useless or confusing. In the high load case, the proximal non-target stimuli have the effect of adding so much extra noise to the units with large receptive fields compared with their signal about the target, that only the smallest receptive fields will be substantially useful. This implies that the distractor will exert little influence. In the low load case, large receptive fields that also include the distractor will be usefully informative about the target, and so the distractor will exert an influence. Note that this happens *automatically* through inference – indeed to make this point starkly, there is no explicit attentional control signal in our model whatsoever, only inference and marginalization.[1]

| load | neutral | | | | incompatible | | | | compatible | | | |
|---|---|---|---|---|---|---|---|---|---|---|---|---|
| | **n** | **t** | **n** | **d** | **n** | **t** | **n** | **d** | **n** | **t** | **n** | **d** |
| **low** | 0 | +c | 0 | 0 | 0 | +c | 0 | -1 | 0 | +c | 0 | +1 |
| **high** | +1 | +c | -1 | 0 | +1 | +c | -1 | -1 | +1 | +c | -1 | +1 |

Table 1: Our version of the task. This table shows 6 out of the 18 conditions. Each display consists of four stimulus positions labelled **n** for the **n**on-targets; **t** for the **t**arget (shown in the table, though not the display, as being boxed); and **d** for the **d**istractor, which is relatively far from the **t**arget. The **t**arget takes the values $\pm c$, where $c$ acts like a contrast; subjects have to report its sign. The **d**istractor can be 0 (neutral) or $\pm 1$; and is compatible if it has the same sign as the **t**arget (and conversely, incompatible). Load is increased by having non-zero **n**on-targets which are spatially *balanced*, with mean 0, so providing no net information about the sign of the **t**arget, but only noise. The 18 conditions come from using $c = \pm 1$ and $c = \pm 0.3$, with the degraded condition ($|c| = 0.3$) only being run for the case of low load, as in figure 1D.

Lavie and de Fockert (2003)'s experiment is rather complicated. Table 1 shows our simplification of it, to a form which is slightly closer to a version of an Eriksen task (Eriksen and Eriksen, 1974) with two optional flankers in known positions on either size of the target (the **n**on-targets) and a farther-flung **d**istractor (the input layer of figure 2A cartoons the spatial arrangement). The **t**arget takes the value $\pm c$; subjects have to report its sign. The **d**istractor can be neutral (0) or have the same sign as (compatible) or a different sign from (incompatible) the **t**arget. In the low load condition, the **n**on-target units are 0; in the high load, one is $+1$; the other is $-1$, making them balanced, but confusing, because they lead to excess noise.

**The generative model**

Table 1 indicates the values determining the various conditions from the perspective of the experimenter. We assume that the subject performs inference about the sign of the **t**arget based on *noisy* observations created by a generative model. In the generative model, the values in table 1 amount to *hidden* structure, which, as in Yu et al. (2008), is mapped and mixed through various receptive fields to provide the noisy input to a Bayesian recognition model. The job of the recognition model is to calculate the posterior probability of the various hidden settings given data, and, by marginalizing (summing) out all the hidden settings apart from the state of the **t**arget, report on its sign.

Figure 2A shows the generative model, indicating the receptive fields (RFs) associated with this mixing. We consider 8 topographically-mapped units, 4 with small RFs covering only a single input (the generative weights are just the identity map); and 4 with large RFs (in which the inputs are mixed together more holistically). Since the **d**istractor is relatively far from the **t**arget and **n**on-target stimuli, the weights associated with its hidden values are lower for the three large RFs mapped to the **t**arget and **n**on-target hidden units; the **t**arget and **n**on-target hidden units have smaller weights to the generated input associated with the **d**istractor. For simplicity, we treat the **d**istractor as equidistant from the **t**arget and **n**on-target input, partially modeling the fact that it can be in different locations. We assume a crude form of signal-dependent noise; it is this that makes the **n**on-target stimuli so devastating.

Figure 2B shows the means and standard deviations arising from the generative model for the 8 units (one per column) for the six conditions in table 1 (rows from top to bottom – low load: neutral, incompatible, compatible; then high load: neutral, incompatible, compatible). For this figure, $c = +1$. The means associated with the small and large RF **t**arget units show the lack of bias from the **n**on-targets in the high-load condition; and for the large RF case, the bias associated with the **d**istractor.

The standard deviations play the most critical role in the model, defining what it means for the **n**on-target stimuli, when present, to make inference difficult. They therefore constitute a key modeling assumption. In the high load case, the units with the large RFs are assumed to have very high standard deviations, coming from a crude form of signal-dependent noise. This captures the relatively uselessness of these large RFs in the high load condition. However, and importantly, their mean values are *unaffected* by the **n**on-target stimuli, since the **n**on-targets are balanced between positive and negative values, preferring neither sign of target.

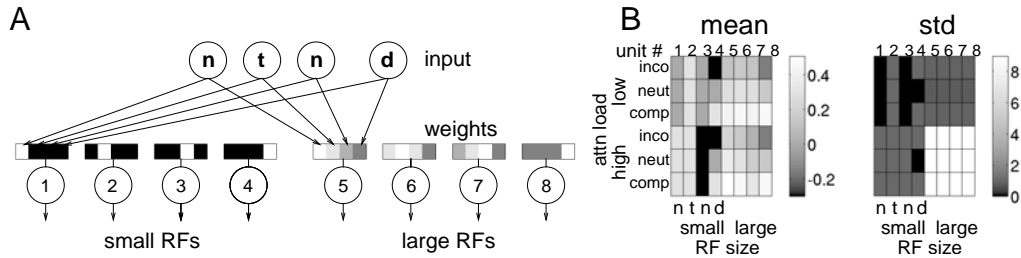

Figure 2: The generative model. A) In the model, the four input units, representing **n**on-targets, the **t**arget and the **d**istractor, are assumed to generate 8 input units which fall into two groups, with small and large receptive fields (RFs). The Hinton diagrams of the weights indicate how the RFs are represented (all weights are positive; the maximum value is 0.3). B) These plots show the means and standard deviations in the generative model associated with the 8 input units for the low and high load cases shown in table 1 (in raster scan order). The means for the large RFs (based on the weights in A) are unaffected by the load; the standard deviations for the units with large receptive fields are much higher in the high load condition. Standard deviations are affected by a coarse form of signal-dependent noise.

In all cases, a new sample from the generative model is provided at each time step; the noise corrupting each of the observed units is assumed to be Gaussian, and independent across units and over time.

**The recognition model**

We build a recognition model based on this generative model. The recognition model is quite similar to a sequential probability ratio test (SPRT; Wald, 1947), except that, as in Yu and Dayan (2005); Yu et al. (2008), it is necessary to perform inference over all the possible values of the hidden variables (all the possible values of the hidden structure[2]), then marginalizing out all the variables apart the the **t**arget itself. We accumulate evidence until a threshold of 0.9 is reached on the probability that the target is either positive or negative (reporting whichever one is more likely). However, to take account of the possibility of erroneous, early, responses, there is also a probability of 0.01 per step of stopping the accumulation and reporting whichever sign of target has a higher probability (guessing randomly if this probability is 0.5). This factor played a critical role in Yu et al. (2008) in generating early responses.

**Results**

Figure 3 shows the results of inference based on the model. For each of the conditions, figure 3A shows the reaction times in the form of the mean number of steps to a choice. Here, as in the data in Lavie and de Fockert (2003), the RTs are averaged only over cases in which the model got the answer correct. However, figure 3B shows the percentage correct answers in each condition; the errors are relatively rare, and so the RTs plots look identical. The datapoints are averages over more than $35,000$ samples (depending on the actual error rates) and so the errorbars are too small to see.

Comparing figure 3A with the data in figure 1D, it is apparent that the main trends in the data are closely captured. This general pattern of results is robust to many different parameter values; though it is possible (by reducing $c$) to make inference take very much longer still in the degraded low load condition whilst maintaining and boosting the effect of high load. The error probabilities in figure 3B indicate that the pattern of RTs is not accounted for by a tradeoff between speed and accuracy.

The three characteristics of these data described above are explained in the model as:

1. In the low load case, the lack of **n**on-targets means that the inputs based on the large RFs are usefully informative about the **t**arget, and therefore automatically play a key role in posterior inference. Since these inputs are also influenced by the **d**istractor, there is an RT

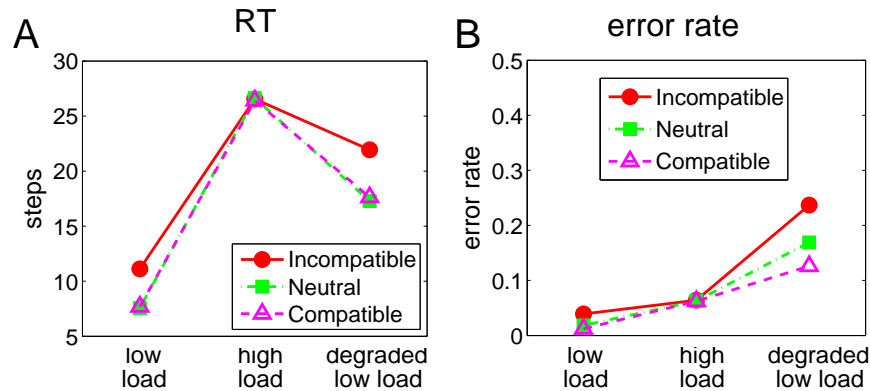

Figure 3: Results. A) Mean RTs (steps of inference) for correct choices in each of the 9 cases (since the **t**arget is equally often positive and negative, we averaged over these cases. Here, the threshold on the (marginalized) probability was $0.9$, and there was a probability of $0.01$ per step that inference would terminate early with whichever response was more probable. B) Error probabilities for the same conditions showing the lack of a speed-accuracy trade-off. All points are averages over more than 35000 points, and so errorbars would be too small to see.

cost in the face of incompatibility. However, in the high load case, the **n**on-target stimuli are closer to the target and exert substantial influence over the noise corrupting the large RF units associated with it (and no net signal). This makes these large RF units relatively poor sources of information about the target. Thus the smaller RF units are relied upon instead, which are not affected by the **d**istractor.

2. Rather as suggested in Norman and Bobrow (1975); Lavie and de Fockert (2003): in the data-poor case of the degraded input, it is particularly important to take advantage of information from the large RFs, to make inferences about the **t**arget; therefore the **d**istractor exerts a large influence over **t**arget processing.

3. The compatible **d**istractor is helpful to a lesser extent than the incompatible one is harmful, for a couple of reasons. First, there is a ceiling effect for the former coming from the non-linearity of an effective sigmoid function that arises in turning log likelihood ratios into probabilities. Second, compared with a neutral **d**istractor, the compatible **d**istractor increases the (signal-dependent) noise associated with the units with large RFs, reducing their informativeness about the target.

## 4 Discussion

In this paper, we have shown how to account for key results used to argue for an attentional load hypothesis. Our model involves simple Bayesian inference based on a generative process recognizing the existence of small and large receptive fields. The attentional load hypothesis suggests that when little attention is required to solve the set task, inputs associated with **d**istractor stimuli *leak* through with little attenuation, and so cause disruption; when the task is difficult, attention is totally occupied with the set task, leaving nothing left over. By contrast, we have suggested that an inferential model taking advantage of all the information in the input will show exactly the same characteristic, with the key issue being whether the units with large RFs, which include the **d**istractor, are rendered useless by the **n**on-target stimuli that make for the high load in the first place. The advantage of this version of an attenuation theory (Treisman, 1960, 1969) is that it obviates the requirement to appeal to an inexplicable inefficiency, over and above the existence of units with large RFs, and indeed relates this set of selective attentional tasks to the wide range of other accounts of probabilistically-correct sensory inference.

One key characteristic of this model (shared with, among others, Yu et al., 2008) is that the form of selection it considers is an *output* of inference rather than an *input* into it. That is, the model does not employ an explicit attentional mechanism in inference which has the capacity to downplay some input units over others. The model does know the location of the target, and focuses all its resources

on it; but there is no further way of boosting or suppressing some RFs compared with others. Most of the substantial results on the neuroscience of selective attention (*eg* Moran and Desimone, 1985; Desimone and Duncan, 1995; Reynolds and Chelazzi, 2004) study the focusing process, rather than the post-focus information integration that we have looked at; the forms of attention at play in the load-related tasks we have discussed are somewhat orthogonal. It would be interesting to design neurophysiological experiments to probe the form of online selection at work in the attentional load tasks.

The difference between the present model and the spatial version of Yu et al. (2008) is that the model here includes RFs of different sizes, whereas in that model, the **d**istractors were always close to the target. Further, the two neutral conditions here (no **d**istractor, and low load) were not modeled in the earlier study. Yu et al. (2008) suggested that the anterior cingulate might monitor conflict between the cases of compatible and incompatible **d**istractors as part of an approximate inference strategy. That seems most unlikely here, since the conflict would have to be between the multidimensional collection of hidden nuisance variables (notably the cross product between the states of the **n**on-targets and the state of the **d**istractor), which seems implausibly complicated.

The assumptions of large RFs and their high standard deviations in the high load condition are certainly rather simplistic. However, (a) RFs in inferotemporal cortex are indeed very large, allowing for the possibility of **d**istractor interference in the low load condition; and (b) even under the attentional load hypothesis, the only reason that an unattenuated **d**istractor stimulus would interfere with target processing is that there is something in common about them, since it is known that there is more to the effects of **d**istractors than just competition at the stage of the actual responses (Driver, 2001). Further, the assumption that the inputs with large RFs have high standard deviations in the high load condition is a most straightforward way to capture the essential effect of the **n**on-target stimuli in disrupting target processing in a way that forces a more stringent attentional effect associated with the use of the small RFs.

The attentional load theory has been applied to many tasks (including the regular Eriksen task, Eriksen and Eriksen, 1974) as well as the one here. However, it would be good to extend the current model to match the experimental circumstances in Lavie and de Fockert (2003) more faithfully. Perhaps the most significant lacuna is that, as in the Eriksen task, we assumed that the subjects knew the location of the target in the stimulus array, whereas in the real experiment, this had to be inferred from the letters in the circle of targets close to fixation (figure 1A). Modeling this would effectively require a more complex collection of letter-based RFs, together with a confusion matrix associated with the perceptual similarities of letters. This induces a search problem, more like the one studied by Mozer and Baldwin (2008), except, again, multiple sizes of RFs would play a critical role. It would also be worth extending the current model to the much wider range of other tasks used to explore the effects of attentional load (such as Forster and Lavie, 2008).

In conclusion, we have suggested a particular rationale for an attenuation theory of attention, which puts together the three tasks suggested at the outset for dichotic listening. Inputs should automatically be attenuated to the extent that they do not bear on (or, worse, are confusing with respect to) a task. The key resource limitation is the restricted number, and therefore, the necessarily broad tuning of RFs; the normative response to his makes attenuation and combination kissing cousins.

### Acknowledgements

I am most grateful to Louise Whiteley for helpful comments and to her and Nillie Lavie for discussions. Funding came from the Gatsby Charitable Foundation.

## Footnotes

[1]Note that Lavie and de Fockert (2003) chose the conditions in the experiment at random, so many forms of top-down selection would not be possible.

[2]In fact, also including the possibility of a degraded high-load case

# References

Broadbent, D. (1958). *Perception and communication*. OUP, Oxford, England.

Cherry, E. (1953). Some experiments on the recognition of speech with one and with two ears. *Journal of the Acoustical Society of America*, 25:975–979.

Dayan, P. and Zemel, R. (1999). Statistical models and sensory attention. In *ICANN 1999*, volume 2. IEE.

Desimone, R. and Duncan, J. (1995). Neural mechanisms of selective visual attention. *Annu Rev Neurosci*, 18:193–222.

Deutsch, J. A. and Deutsch, D. (1963). Attention: Some theoretical considerations. *Psychol Rev*, 70:80–90.

Driver, J. (2001). A selective review of selective attention research from the past century. *Br J Psychol*, 92 Part 1:53–78.

Duncan, J. (1980). The locus of interference in the perception of simultaneous stimuli. *Psychol Rev*, 87(3):272–300.

Eriksen, B. and Eriksen, C. (1974). Effects of noise-letters on identification of a target letter in a nonsearch task. *Perception & Psychophysics*, 16:143–149.

Eriksen, C. W. and St. James, J. D. (1986). Visual attention within and around the field of focal attention: a zoom lens model. *Percept Psychophys*, 40(4):225–240.

Forster, S. and Lavie, N. (2008). Failures to ignore entirely irrelevant distractors: the role of load. *J Exp Psychol Appl*, 14(1):73–83.

Itti, L. and Koch, C. (2001). Computational modelling of visual attention. *Nat Rev Neurosci*, 2(3):194–203.

James, W. (1890/1950). *The Principles of Psychology*. Dover, New York, NY.

Lavie, N. (2005). Distracted and confused?: selective attention under load. *Trends Cogn Sci*, 9(2):75–82.

Lavie, N. and de Fockert, J. W. (2003). Contrasting effects of sensory limits and capacity limits in visual selective attention. *Percept Psychophys*, 65(2):202–212.

Lavie, N. and Tsal, Y. (1994). Perceptual load as a major determinant of the locus of selection in visual attention. *Percept Psychophys*, 56(2):183–197.

Mackay, D. (1973). Aspects of the theory of comprehension, memory and attention. *Quarterly Journal of Experimental Psychology,*, 25:22–40.

Moran, J. and Desimone, R. (1985). Selective attention gates visual processing in the extrastriate cortex. *Science*, 229(4715):782–784.

Mozer, M. and Baldwin, D. (2008). Experience-guided search: A theory of attentional control. In Platt, J., Koller, D., Singer, Y., and Roweis, S., editors, *Advances in Neural Information Processing Systems 20*, pages 1033–1040. MIT Press, Cambridge, MA.

Navalpakkam, V. and Itti, L. (2006). Optimal cue selection strategy. In Weiss, Y., Schölkopf, B., and Platt, J., editors, *Advances in Neural Information Processing Systems 18*, pages 987–994. MIT Press, Cambridge, MA.

Norman, D. and Bobrow, D. (1975). On Data-limited and Resource-limited Processes. *Cognitive Psychology*, 7(1):44–64.

Palmer, J. (1994). Set-size effects in visual search: the effect of attention is independent of the stimulus for simple tasks. *Vision Res*, 34(13):1703–1721.

Paschler, H. (1998). *The Psychology of Attention*. MIT Press, Cambridge, MA.

Rao, R. P. N., Olshausen, B. A., and Lewicki, M. S., editors (2002). *Probabilistic Models of the Brain*. MIT Press, Cambridge, MA.

Reynolds, J. H. and Chelazzi, L. (2004). Attentional modulation of visual processing. *Annu Rev Neurosci*, 27:611–647.

Shaw, M. (1982). Attending to multiple sources of information. *Cognitive Psychology*, 14:353–409.

Treisman, A. M. (1960). Contextual cues in selective listening. *Quarterly Journal of Experimental Psychology*, 12:242–248.

Treisman, A. M. (1969). Strategies and models of selective attention. *Psychol Rev*, 76(3):282–299.

Wald, A. (1947). *Sequential Analysis*. Wiley, New York.

Whiteley, L. and Sahani, M. (2008). Attention resolves the effects of a computational bottleneck: modelling binding, precueing, and task-driven bias. In *COSYNE 2008*, pages I–98.

Yu, A., Dayan, P., and Cohen, J. (2008). Bayesian account of attentional control. *Journal of Experimental Psychology: Human Percept Psychophys*, in press.

Yu, A. J. and Dayan, P. (2005). Inference, attention, and decision in a bayesian neural architecture. In Saul, L. K., Weiss, Y., and Bottou, L., editors, *Advances in Neural Information Processing Systems 17*, pages 1577–1584. MIT Press, Cambridge, MA.

Zhaoping, L. (2006). Theoretical understanding of the early visual processes by data compression and data selection. *Network*, 17(4):301–334.
